# Approximate inference using planar graph decomposition

**Amir Globerson    Tommi Jaakkola**
Computer Science and Artificial Intelligence Laboratory
Massachusetts Institute of Technology
Cambridge, MA 02139
gamir,tommi@csail.mit.edu

## Abstract

A number of exact and approximate methods are available for inference calculations in graphical models. Many recent approximate methods for graphs with cycles are based on tractable algorithms for tree structured graphs. Here we base the approximation on a different tractable model, planar graphs with binary variables and pure interaction potentials (no external field). The partition function for such models can be calculated exactly using an algorithm introduced by Fisher and Kasteleyn in the 1960s. We show how such tractable planar models can be used in a decomposition to derive upper bounds on the partition function of non-planar models. The resulting algorithm also allows for the estimation of marginals. We compare our planar decomposition to the tree decomposition method of Wainwright *et. al.*, showing that it results in a much tighter bound on the partition function, improved pairwise marginals, and comparable singleton marginals.

Graphical models are a powerful tool for modeling multivariate distributions, and have been successfully applied in various fields such as coding theory and image processing. Applications of graphical models typically involve calculating two types of quantities, namely marginal distributions, and MAP assignments. The evaluation of the model partition function is closely related to calculating marginals [12]. These three problems can rarely be solved exactly in polynomial time, and are provably computationally hard in the general case [1]. When the model conforms to a tree structure, however, all these problems can be solved in polynomial time. This has prompted extensive research into *tree based* methods. For example, the junction tree method [6] converts a graphical model into a tree by clustering nodes into cliques, such that the graph over cliques is a tree. The resulting maximal clique size (cf. *tree width*) may nevertheless be prohibitively large.

Wainwright *et. al.* [9, 11] proposed an approximate method based on trees known as tree reweighting (TRW). The TRW approach decomposes the potential vector of a graphical model into a mixture over spanning trees of the model, and then uses convexity arguments to bound various quantities, such as the partition function. One key advantage of this approach is that it provides bounds on partition function value, a property which is not shared by approximations based on Bethe free energies [13].

In this paper we focus on a different class of tractable models: planar graphs. A graph is called planar if it can be drawn in the plane without crossing edges. Works in the 1960s by physicists Fisher [5] and Kasteleyn [7], among others, have shown that the partition function for planar graphs may be calculated in polynomial time. This, however, is true under two key restrictions. One is that the variables $x_i$ are binary. The other is that the interaction potential depends only on $x_i x_j$ (where $x_i \in \{\pm 1\}$), and not on their individual values (i.e., the zero external field case).

Here we show how the above method can be used to obtain upper bounds on the partition function for *non-planar* graphs. As in TRW, we decompose the potential of a non-planar graph into a sum

over spanning planar models, and then use a convexity argument to obtain an upper bound on the log partition function. The bound optimization is a convex problem, and can be solved in polynomial time. We compare our method with TRW on a planar graph with an external field, and show that it performs favorably with respect to both pairwise marginals and the bound on the partition function, and the two methods give similar results for singleton marginals.

## 1  Definitions and Notations

Given a graph $G$ with $n$ vertices and a set of edges $E$, we are interested in pairwise Markov Random Fields (MRF) over the graph $G$. A pairwise MRF [13] is a multivariate distribution over variables $\mathbf{x} = \{x_1, \ldots, x_n\}$ defined as

$$p(\mathbf{x}) = \frac{1}{Z} e^{\sum_{ij \in E} f_{ij}(x_i, x_j)} \tag{1}$$

where $f_{ij}$ are a set of $|E|$ functions, or interaction potentials, defined over pairs of variables. The partition function is defined as $Z = \sum_{\mathbf{x}} e^{\sum_{ij \in E} f_{ij}(x_i, x_j)}$. Here we will focus on the case where $x_i \in \{\pm 1\}$. Furthermore, we will be interested in interaction potentials which only depend on agreement or disagreement between the signs of their variables. We define those by

$$f(x_i, x_j) = \frac{1}{2}\theta_{ij}(1 + x_i x_j) = \theta_{ij}\mathcal{I}(x_i = x_j) \tag{2}$$

so that $f_{ij}(x_i, x_j)$ is zero if $x_i \neq x_j$ and $\theta_{ij}$ if $x_i = x_j$. The model is then defined via the set of parameters $\theta_{ij}$. We use $\boldsymbol{\theta}$ to denote the vector of parameters $\theta_{ij}$, and denote the partition function by $Z(\boldsymbol{\theta})$ to highlight its dependence on these parameters.

A graph $G$ is defined as planar if it can be drawn in the plane without any intersection of edges [4]. With some abuse of notation, we define $E$ as the set of line segments in $\Re^2$ corresponding to the edges in the graph. The regions of $\Re^2 \setminus E$ are defined as the *faces* of the graph. The face which corresponds to an unbounded region is called the *external face*. Given a planar graph $G$, its *dual* graph $G^*$ is defined in the following way: the vertices of $G^*$ correspond to faces of $G$, and there is an edge between two vertices in $G^*$ iff the two corresponding faces in $G$ share an edge. If the graph $G$ is weighted, the weight on an edge in $G^*$ is the weight on the edge shared by the corresponding faces in $G$.

A *plane triangulation* of a planar graph $G$ is obtained from $G$ by adding edges such that all the faces of the resulting graph have exactly three vertices. Thus a plane triangulated graph has a dual where all vertices have degree three. It can be shown that every plane graph can be plane triangulated [4].

We shall also need the notion of a *perfect matching* on a graph. A perfect matching on a graph $G$ is defined as a set of edges $H \subseteq E$ such that every vertex in $G$ has exactly one edge in $H$ incident on it. If the graph is weighted, the weight of the matching is defined as the product of the weights of the edges in the matching.

Finally, we recall the definition of a *marginal polytope* of a graph [12]. Consider an MRF over a graph $G$ where $f_{ij}$ are given by Equation 2. Denote the probability of the event $\mathcal{I}(x_i = x_j)$ under $p(\mathbf{x})$ by $\tau_{ij}$. The *marginal polytope* of $G$, denoted by $\mathcal{M}(G)$, is defined as the set of values $\tau_{ij}$ that can be obtained under some assignment to the parameters $\theta_{ij}$. For a general graph $G$ the polytope $\mathcal{M}(G)$ cannot be described using a polynomial number of inequalities. However, for planar graphs, it turns out that a set of $O(n^3)$ constraints, commonly referred to as *triangle inequalities*, suffice to describe $\mathcal{M}(G)$ (see [3] page 434). The triangle inequalities are defined by [1]

$$\text{TRI}(n) = \{\tau_{ij} : \tau_{ij} + \tau_{jk} - \tau_{ik} \leq 1, \tau_{ij} + \tau_{jk} + \tau_{ik} \geq 1, \forall i, j, k \in \{1, \ldots, n\}\} \tag{3}$$

Note that the above inequalities actually contain variables $\tau_{ij}$ which do not correspond to edges in the original graph $G$. Thus the equality $\mathcal{M}(G) = \text{TRI}(n)$ should be understood as referring only to the values of $\tau_{ij}$ that correspond to edges in the graph. Importantly, the values of $\tau_{ij}$ for edges *not* in the graph need not be valid marginals for any MRF. In other words $\mathcal{M}(G)$ is a projection of TRI$(n)$ on the set of edges of $G$. It is well known that the marginal polytope for trees is described via pairwise constraints. It is thus interesting that for planar graphs, it is triplets, rather than pairwise

constraints, that characterize the polytope. In this sense, planar graphs and trees may be viewed as a hierarchy of polytope complexity classes. It remains an interesting problem to characterize other structures in this hierarchy and their related inference algorithms.

## 2 Exact calculation of partition function using perfect matching

The seminal works of Kasteleyn [7] and Fisher [5] have shown how one can calculate the partition function for a binary MRF over a planar graph with pure interaction potentials. We briefly review Fisher's construction, which we will use in what follows. Our interpretation of the method differs somewhat from that of Fisher, but we believe it is more straightforward. The key idea in calculating the partition function is to convert the summation over values of $\mathbf{x}$ to the problem of calculating the sum of weights of all perfect matchings in a graph constructed from $G$, as shown below.

In this section, we consider weighted graphs (graphs with numbers assigned to their edges). For the graph $G$ associated with the pairwise MRF, we assign weights $w_{ij} = e^{2\theta_{ij}}$ to the edges. The first step in the construction is to plane triangulate the graph $G$. Let us call the resulting graph $G_{\mathrm{T}}$. We define an MRF on $G_{\mathrm{T}}$ by assigning a parameter $\theta_{ij} = 0$ to the edges that have been added to $G$, and the corresponding weight $w_{ij} = 1$. Thus $G_{\mathrm{T}}$ essentially describes the same distribution as $G$, and therefore has the same partition function. We can thus restrict our attention to calculating the partition function for the MRF on $G_{\mathrm{T}}$.

As a first step in calculating a partition function over $G_{\mathrm{T}}$, we introduce the following definition: a set of edges $\hat{E}$ in $G_{\mathrm{T}}$ is an *agreement edge set* (or AES) if for every triangle face $F$ in $G_{\mathrm{T}}$ one of the following holds: The edges in $F$ are all in $\hat{E}$, or exactly one of the edges in $F$ is in $\hat{E}$. The weight of a set $\hat{E}$ is defined as the product of the weights of the edges in $\hat{E}$.

It can be shown that there exists a bijection between pairs of assignments $\{\mathbf{x}, -\mathbf{x}\}$ and agreement edge sets. The mapping from $\mathbf{x}$ to an edge set is simply the set of edges such that $x_i = x_j$. It is easy to see that this is an agreement edge set. The reverse mapping is obtained by finding an assignment $\mathbf{x}$ such that $x_i = x_j$ iff the corresponding edge is in the agreement edge set. The existence of this mapping can be shown by induction on the number of (triangle) faces.

The contribution of a given assignment $\mathbf{x}$ to the partition function is $e^{\sum_{ij \in E} \theta_{ij} \mathcal{I}(x_i = x_j)}$. If $\mathbf{x}$ corresponds to an AES denoted by $\hat{E}$ it is easy to see that

$$e^{\sum_{ij \in E} \theta_{ij} \mathcal{I}(x_i = x_j)} = e^{-\sum_{ij \in E} \theta_{ij}} e^{\sum_{ij \in \hat{E}} 2\theta_{ij}} = c \, e^{\sum_{ij \in \hat{E}} 2\theta_{ij}} = c \prod_{ij \in \hat{E}} w_{ij} \qquad (4)$$

where $c = e^{-\sum_{ij \in E} \theta_{ij}}$. Define the superset $\Lambda$ as the set of agreement edge sets. The above then implies that $Z(\boldsymbol{\theta}) = 2c \sum_{\hat{E} \in \Lambda} \prod_{ij \in \hat{E}} w_{ij}$, and is thus proportional to the sum of AES weights. To sum over agreement edge sets, we use the following elegant trick introduced by Fisher [5]. Construct a new graph $G_{\mathrm{PM}}$ from the *dual* of $G_{\mathrm{T}}$ by introducing new vertices and edges according to the following rule: Replace each original vertex with three vertices that are connected to each other, and assign a weight of one to the new edges. Next, consider the three neighbors of the original vertex [2]. Connect each of the three new vertices to one of these three neighbors, keeping the original weights on these edges. The transformation is illustrated in Figure 1.

The new graph $G_{\mathrm{PM}}$ has $O(3n)$ vertices, and is also planar. It can be seen that there is a one to one correspondence between perfect matchings in $G_{\mathrm{PM}}$ and agreement edge sets in $G_{\mathrm{T}}$. Define $\Omega$ to be the set of perfect matchings in $G_{\mathrm{PM}}$. Then $Z(\boldsymbol{\theta}) = 2c \sum_{M \in \Omega} \prod_{ij \in M} w_{ij}$ where we have used the fact that all the new weights have a value of one. Thus, the partition function is a sum over the weights of perfect matchings in $G_{\mathrm{PM}}$. Finally, we need a way of summing over the weights of the set of perfect matchings in a graph. Kasteleyn [7] proved that for a planar graph $G_{\mathrm{PM}}$, this sum may be obtained using the following sequence of steps:

- Direct the edges of the graph $G_{\mathrm{PM}}$ such that for every face (except possibly the external face), the number of edges on its perimeter oriented in a clockwise manner is odd. Kasteleyn showed that such a so called Pfaffian orientation may be constructed in polynomial time for a planar graph (see also [8] page 322).

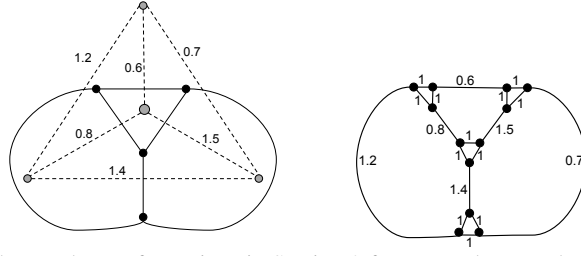

Figure 1: Illustration of the graph transformations in Section 2 for a complete graph with four vertices. Left panel shows the original weighted graph (dotted edges and grey vertices) and its dual (solid edges and black vertices). Right panel shows the dual graph with each vertex replaced by a triangle (the graph $G_{\text{PM}}$ in the text). Weights for dual graph edges correspond to the weights on the original graph.

- Define the matrix $P(G_{\text{PM}})$ to be a skew symmetric matrix such that $P_{ij} = 0$ if $ij$ is not an edge, $P_{ij} = w_{ij}$ if the arrow on edge $ij$ runs from $i$ to $j$ and $P_{ij} = -w_{ij}$ otherwise.

- The sum over weighted matchings can then be shown to equal $\sqrt{|P(G_{\text{PM}})|}$.

The partition function is thus given by $Z(\boldsymbol{\theta}) = 2c\sqrt{|P(G_{\text{PM}})|}$. To conclude this section we reiterate the following two key points: the partition function of a binary MRF over a planar graph with interaction potentials as in Equation 2 may be calculated in polynomial time by calculating the determinant of a matrix of size $O(3n)$. An important outcome of this result is that the functional relation between $Z(\boldsymbol{\theta})$ and the parameters $\theta_{ij}$ is known, a fact we shall use in what follows.

## 3  Partition function bounds via planar decomposition

Given a non-planar graph $G$ over binary variables with a vector of interaction potentials $\boldsymbol{\theta}$, we wish to use the exact planar computation to obtain a bound on the partition function of the MRF on $G$. We assume for simplicity that the potentials on the MRF for $G$ are given in the form of Equation 2. Thus, $G$ violates the assumptions of the previous section only in its non-planarity. Define $G^{(r)}$ as a set of spanning planar subgraphs of $G$, i.e., each graph $G^{(r)}$ is planar and contains all the vertices of $G$ and some its edges. Denote by $m$ the number of such graphs. Introduce the following definitions:

- $\boldsymbol{\theta^{(r)}}$ is a set of parameters on the edges of $G^{(r)}$, and $\theta_{ij}^{(r)}$ is an element in this set. $Z(\boldsymbol{\theta^{(r)}})$ is the partition function of the MRF on $G^{(r)}$ with parameters $\boldsymbol{\theta^{(r)}}$.

- $\hat{\boldsymbol{\theta}}^{(r)}$ is a set of parameters on the edges of $G$ such that if edge $(ij)$ is in $G^{(r)}$ then $\hat{\theta}_{ij}^{(r)} = \theta_{ij}^{(r)}$, and otherwise $\hat{\theta}_{ij}^{(r)} = 0$.

Given a distribution $\rho(r)$ on the graphs $G^{(r)}$ (i.e., $\rho(r) \geq 0$ for $r = 1, \ldots, m$ and $\sum_r \rho(r) = 1$), assume that the parameters for $G^{(r)}$ are such that

$$\boldsymbol{\theta} = \sum_r \rho(r)\hat{\boldsymbol{\theta}}^{(r)} \tag{5}$$

Then, by the convexity of the log partition function, as a function of the model parameters, we have

$$\log Z(\boldsymbol{\theta}) \leq \sum_r \rho(r) \log Z(\boldsymbol{\theta^{(r)}}) \equiv f(\boldsymbol{\theta}, \rho, \boldsymbol{\theta^{(r)}}) \tag{6}$$

Since by assumption the graphs $G^{(r)}$ are planar, this bound can be calculated in polynomial time. Since this bound is true for any set of parameters $\boldsymbol{\theta^{(r)}}$ which satisfies the condition in Equation 5 and for any distribution $\rho(r)$, we may optimize over these two variables to obtain the tightest bound possible. Define the optimal bound for a fixed value of $\rho(r)$ by $g(\rho, \boldsymbol{\theta})$ (optimization is w.r.t. $\boldsymbol{\theta^{(r)}}$)

$$g(\rho, \boldsymbol{\theta}) = \min_{\boldsymbol{\theta^{(r)}}:\sum \rho(r)\hat{\boldsymbol{\theta}}^{(r)}=\boldsymbol{\theta}} f(\boldsymbol{\theta}, \rho, \boldsymbol{\theta^{(r)}}) \tag{7}$$

Also, define the optimum of the above w.r.t. $\rho$ by $h(\boldsymbol{\theta})$.

$$h(\boldsymbol{\theta}) = \min_{\rho(r) \geq 0, \sum \rho(r) = 1} g(\boldsymbol{\theta}, \rho) \tag{8}$$

Thus, $h(\boldsymbol{\theta})$ is the optimal upper bound for the given parameter vector $\boldsymbol{\theta}$. In the following section we argue that we can in fact find the global optimum of the above problem.

## 4 Globally Optimal Bound Optimization

First consider calculating $g(\rho, \boldsymbol{\theta})$ from Equation 7. Note that since $\log Z(\boldsymbol{\theta}^{(r)})$ is a convex function of $\boldsymbol{\theta}^{(r)}$, and the constraints are linear, the overall optimization is convex and can be solved efficiently. In the current implementation, we use a projected gradient algorithm [2]. The gradient of $f(\boldsymbol{\theta}, \rho, \boldsymbol{\theta}^{(r)})$ w.r.t. $\boldsymbol{\theta}^{(r)}$ is given by

$$\frac{\partial f(\boldsymbol{\theta}, \rho, \boldsymbol{\theta}^{(r)})}{\partial \theta_{ij}^{(r)}} = \rho(r)\left(1 + e^{\theta_{ij}^{(r)}}\left[P^{-1}(G_{\text{PM}}^{(r)})\right]_{k(i,j)}\text{Sign}(P_{k(i,j)}(G_{\text{PM}}^{(r)}))\right) \tag{9}$$

where $k(i,j)$ returns the row and column indices of the element in the upper triangular matrix of $P(G_{\text{PM}}^{(r)})$, which contains the element $e^{2\theta_{ij}^{(r)}}$.

Since the optimization in Equation 7 is convex, it has an equivalent convex dual. Although we do not use this dual for optimization (because of the difficulty of expressing the entropy of planar models solely in terms of triplet marginals), it nevertheless allows some insight into the structure of the problem. The dual in this case is closely linked to the notion of the marginal polytope defined in Section 1. Using a derivation similar to [11], we arrive at the following characterization of the dual

$$g(\rho, \boldsymbol{\theta}) = \max_{\tau \in \text{TRI}_{(n)}} \boldsymbol{\theta} \cdot \tau + \sum_r \rho(r) H(\boldsymbol{\theta}^{(r)}(\tau)) \tag{10}$$

where $\boldsymbol{\theta}^{(r)}(\tau)$ denotes the parameters of an MRF on $G^{(r)}$ such that its marginals are given by the restriction of $\tau$ to the edges of $G^{(r)}$, and $H(\boldsymbol{\theta}^{(r)}(\tau))$ denotes the entropy of the MRF over $G^{(r)}$ with parameters $\boldsymbol{\theta}^{(r)}(\tau)$. The maximized function in Equation 10 is linear in $\rho$ and thus $g(\rho, \boldsymbol{\theta})$ is a pointwise maximum over (linear) convex functions in $\rho$ and is thus convex in $\rho$. It therefore has no local minima. Denote by $\boldsymbol{\theta}_{min}^{(r)}(\rho)$ the set of parameters that minimizes Equation 7 for a given value of $\rho$. Using a derivation similar to that in [11], the gradient of $g(\rho, \boldsymbol{\theta})$ can be shown to be

$$\frac{\partial g(\rho, \boldsymbol{\theta})}{\partial \rho(r)} = H(\boldsymbol{\theta}_{min}^{(r)}(\rho)) \tag{11}$$

Since the partition function for $G^{(r)}$ can be calculated efficiently, so can the entropy. We can now summarize the algorithm for calculating $h(\boldsymbol{\theta})$

- Initialize $\rho_0$. Iterate:
  - For $\rho_t$, find $\boldsymbol{\theta}^{(r)}$ which solves the minimization in Equation 7.
  - Calculate the gradient of $g(\rho, \boldsymbol{\theta})$ at $\rho_t$ using the expression in Equation 11
  - Update $\rho_{t+1} = \rho_t + \alpha \mathbf{v}$ where $\mathbf{v}$ is a feasible search direction calculated from the gradient of $g(\rho, \boldsymbol{\theta})$ and the simplex constraints on $\rho$. The step size $\alpha$ is calculated via an Armijo line search.
  - Halt when the change in $g(\rho, \boldsymbol{\theta})$ is smaller than some threshold.

Note that the minimization w.r.t. $\boldsymbol{\theta}^{(r)}$ is not very time consuming since we can initialize it with the minimum from the previous step, and thus only a few iterations are needed to find the new optimum, provided the change in $\rho$ is not too big. The above algorithm is guaranteed to converge to a global optimum of $\rho$ [2], and thus we obtain the tightest possible upper bound on $Z(\boldsymbol{\theta})$ given our planar graph decomposition.

The procedure described here is asymmetric w.r.t. $\rho$ and $\boldsymbol{\theta}^{(r)}$. In a symmetric formulation the minimizing gradient steps could be carried out jointly or in an alternating sequence. The symmetric formulation can be obtained by decoupling $\rho$ and $\boldsymbol{\theta}^{(r)}$ in the bi-linear constraint $\sum \rho(r)\hat{\boldsymbol{\theta}}^{(r)} = \boldsymbol{\theta}$.

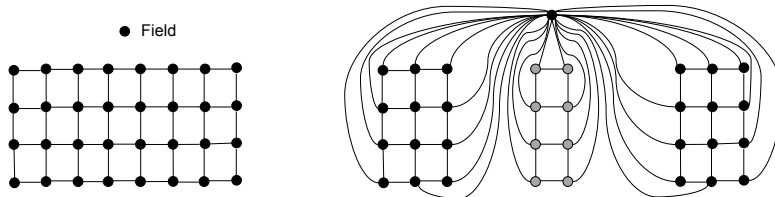

Figure 2: Illustration of planar subgraph construction for a rectangular lattice with external field. Original graph is shown on the left. The field vertex is connected to all vertices (edges not shown). The graph on the right results from *isolating* the $4^{th}, 5^{th}$ columns of the original graph (shown in grey), and connecting the field vertex to the external vertices of the three disconnected components. Note that the resulting graph is planar.

Specifically, we introduce $\tilde{\boldsymbol{\theta}}^{(r)} = \boldsymbol{\theta}^{(r)} \rho(r)$ and perform the optimization w.r.t. $\rho$ and $\tilde{\boldsymbol{\theta}}^{(r)}$. It can be shown that a stationary point of $f(\boldsymbol{\theta}, \rho, \tilde{\boldsymbol{\theta}}^{(r)})$ with the relevant (de-coupled) constraint is equivalent to the procedure described above. The advantage of this approach is that the exact minimization w.r.t $\boldsymbol{\theta}^{(r)}$ is not required before modifying $\rho$. Our experiments have shown, however, that the methods take comparable times to converge, although this may be a property of the implementation.

## 5  Estimating Marginals

The optimization problem as defined above minimizes an upper bound on the partition function. However, it may also be of interest to obtain estimates of the marginals of the MRF over $G$. To obtain marginal estimates, we follow the approach in [11]. We first characterize the optimum of Equation 7 for a fixed value of $\rho$. Deriving the Lagrangian of Equation 7 w.r.t. $\boldsymbol{\theta}^{(r)}$ we obtain the following characterization of $\boldsymbol{\theta}_{min}^{(r)}(\rho)$:

Marginal Optimality Criterion:  For any two graphs $G^{(r)}, G^{(s)}$ such that the edge $(ij)$ is in both graphs, the optimal parameter vector satisfies $\tau_{ij}(\boldsymbol{\theta}_{min}^{(r)}(\rho)) = \tau_{ij}(\boldsymbol{\theta}_{min}^{(s)}(\rho))$.

Thus, the optimal set of parameters for the graphs $G^{(r)}$ is such that every two graphs agree on the marginals of all the edges they share. This implies that at the optimum, there is a well defined set of marginals over all the edges. We use this set as an approximation to the true marginals.

A different method for estimating marginals uses the partition function bound directly. We first calculate partition function bounds on the sums: $\alpha_i(1) = \sum_{\mathbf{x}:x_i=1} e^{\sum_{ij \in E} f_{ij}(x_i, x_j)}$ and $\alpha_i(-1) = \sum_{\mathbf{x}:x_i=-1} e^{\sum_{ij \in E} f_{ij}(x_i, x_j)}$ and then normalize $\frac{\alpha_i(1)}{\alpha_i(1)+\alpha_i(-1)}$ to obtain an estimate for $p(x_i = 1)$. This method has the advantage of being more numerically stable (since it does not depend on derivatives of $\log Z$). However, it needs to be calculated separately for each variable, so that it may be time consuming if one is interested in marginals for a large set of variables.

## 6  Experimental Evaluation

We study the application of our Planar Decomposition (PDC) method to a binary MRF on a square lattice with an external field. The MRF is given by $p(\mathbf{x}) \propto e^{\sum_{ij \in E} \theta_{ij} x_i x_j + \sum_{i \in V} \theta_i x_i}$ where $V$ are the lattice vertices, and $\theta_i$ and $\theta_{ij}$ are parameters. Note that this interaction does not satisfy the conditions for exact calculation of the partition function, even though the graph is planar. This problem is in fact NP hard [1]. However, it is possible to obtain the desired interaction form by introducing an additional variable $x_{n+1}$ that is connected to all the original variables. Denote the corresponding graph by $G_f$. Consider the distribution $p(\mathbf{x}, x_{n+1}) \propto e^{\sum_{ij \in E} \theta_{ij} x_i x_j + \sum_{i \in V} \theta_{i,n+1} x_i x_{n+1}}$, where $\theta_{i,n+1} = \theta_i$. It is easy to see that any property of $p(\mathbf{x})$ (e.g., partition function, marginals) may be calculated from the corresponding property of $p(\mathbf{x}, x_{n+1})$. The advantage of the latter distribution is that it has the desired interaction form. We can thus apply PDC by choosing planar subgraphs of the non-planar graph $G_f$.

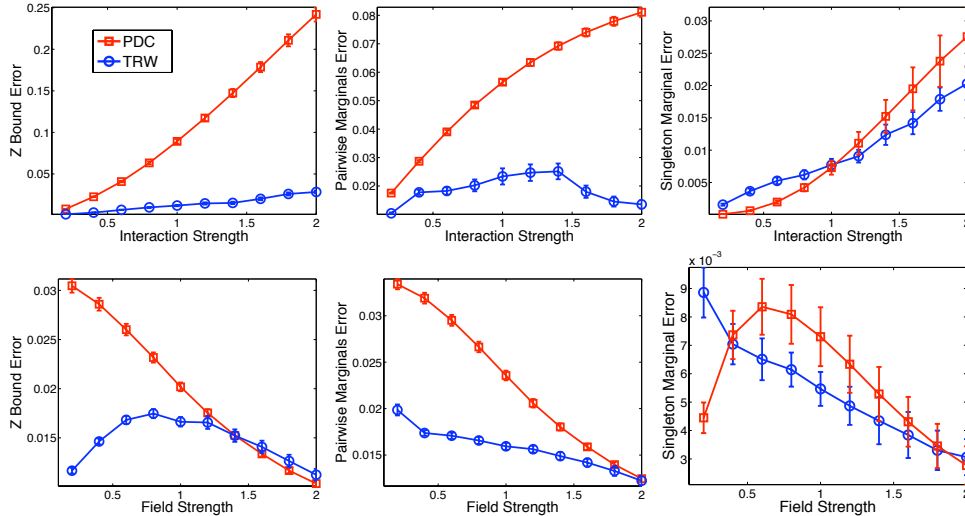

Figure 3: Comparison of the TRW and Planar Decomposition (PDC) algorithms on a $7 \times 7$ square lattice. TRW results shown in red squares, and PDC in blue circles. Left column shows the error in the log partition bound. Middle column is the mean error for pairwise marginals, and right column is the error for the singleton marginal of the variable at the lattice center. Results in upper row are for field parameters drawn from $\mathcal{U}[-0.05, 0.05]$ and various interaction parameters. Results in the lower row are for interaction parameters drawn from $\mathcal{U}[-0.5, 0.5]$ and various field parameters. Error bars are standard errors calculated from 40 random trials.

There are clearly many ways to choose spanning planar subgraphs of $G_f$. Spanning subtrees are one option, and were used in [11]. Since our optimization is polynomial in the number of subgraphs, we preferred to use a number of subgraphs that is linear in $\sqrt{n}$. The key idea in generating these planar subgraphs is to generate disconnected components of the lattice and connect $x_{n+1}$ only to the external vertices of these components. Here we generate three disconnected components by isolating two neighboring columns (or rows) from the rest of the graph, resulting in three components. This is illustrated in Figure 2. To this set of $2\sqrt{n}$ graphs, we add the *independent variables* graph consisting only of edges from the field node to all the other nodes.

We compared the performance of the PDC and TRW methods [3][4] on a $7 \times 7$ lattice . Since the exact partition function and marginals can be calculated for this case, we could compare both algorithms to the true values. The MRF parameters were set according to the two following scenarios: 1) Varying Interaction - The field parameters $\theta_i$ were drawn uniformly from $\mathcal{U}[-0.05, 0.05]$, and the interaction $\theta_{ij}$ from $\mathcal{U}[-\alpha, \alpha]$ where $\alpha \in \{0.2, 0.4, \dots, 2\}$. This is the setting tested in [11]. 2) Varying Field - $\theta_i$ was drawn uniformly from $\mathcal{U}[-\alpha, \alpha]$, where $\alpha \in \{0.2, 0.4, \dots, 2\}$ and $\theta_{ij}$ from $\mathcal{U}[-0.5, 0.5]$.

For each scenario, we calculated the following measures: 1) Normalized log partition error $\frac{1}{49}(\log Z^{alg} - \log Z^{true})$. 2) Error in pairwise marginals $\frac{1}{|E|}\sum_{ij\in E}|p^{alg}(x_i = 1, x_j = 1) - p^{true}(x_i = 1, x_j = 1)|$. Pairwise marginals were calculated jointly using the marginal optimality criterion of Section 5. 3) Error in singleton marginals. We calculated the singleton marginals for the innermost node in the lattice (i.e., coordinate $[3, 3]$), which intuitively should be the most difficult for the planar based algorithm. This marginal was calculated using two partition functions, as explained in Section 5 [5]. The same method was used for TRW. The reported error measure is $|p^{alg}(x_i = 1) - p^{true}(x_i = 1)|$. Results were averaged over 40 random trials.

Results for the two scenarios and different evaluation measures are given in Figure 3. It can be seen that the partition function bound for PDC is significantly better than TRW for almost all parameter settings, although the difference becomes smaller for large field values. Error for the PDC pairwise

marginals are smaller than those of TRW for all parameter settings. For the singleton parameters, TRW slightly outperforms PDC. This is not surprising since the field is modeled by every spanning tree in the TRW decomposition, whereas in PDC not all the structures model a given field.

## 7 Discussion

We have presented a method for using planar graphs as the basis for approximating non-planar graphs such as planar graphs with external fields. While the restriction to binary variables limits the applicability of our approach, it remains relevant in many important applications, such as coding theory and combinatorial optimization. Moreover, it is always possible to convert a non-binary graphical model to a binary one by introducing additional variables. The resulting graph will typically not be planar, even when the original graph over $k-$ary variables is. However, the planar decomposition method can then be applied to this non-planar graph.

The optimization of the decomposition is carried out explicitly over the planar subgraphs, thus limiting the number of subgraphs that can be used in the approximation. In the TRW method this problem is circumvented since it is possible to implicitly optimize over *all* spanning trees. The reason this can be done for trees is that the entropy of an MRF over a tree may be written as a function of its marginal variables. We do not know of an equivalent result for planar graphs, and it remains a challenge to find one. It is however possible to combine the planar and tree decompositions into one single bound, which is guaranteed to outperform the tree or planar approximations alone.

The planar decomposition idea may in principle be applied to bounding the value of the MAP assignment. However, as in TRW, it can be shown that the solution is not dependent on the decomposition (as long as each edge appears in some structure), and the problem is equivalent to maximizing a linear function over the marginal polytope (which can be done in polynomial time for planar graphs). However, such a decomposition may suggest new message passing algorithms, as in [10].

### Acknowledgments

The authors acknowledge support from the Defense Advanced Research Projects Agency (Transfer Learning program). Amir Globerson is also supported by the Rothschild Yad-Hanadiv fellowship. The authors also wish to thank Martin Wainwright for providing his TRW code.

## Footnotes

[1] The definition here is slightly different from that in [3], since here we refer to agreement probabilities, whereas [3] refers to disagreement probabilities. This polytope is also referred to as the *cut polytope*.

[2]Note that in the dual of $G_{\mathrm{T}}$ all vertices have degree three, since $G_{\mathrm{T}}$ is plane triangulated.

[3]TRW and PDC bounds were optimized over both the subgraph parameters and the mixture parameters $\rho$.

[4]In terms of running time, PDC optimization for a fixed value of $\rho$ took about 30 seconds, which is still slower than the TRW message passing implementation.

[5]Results using the marginal optimality criterion were worse for PDC, possibly due to its reduced numerical precision.

## References

[1] F. Barahona. On the computational complexity of ising spin glass models. *J. Phys. A.*, 15(10):3241–3253, 1982.

[2] D. P. Bertsekas, editor. *Nonlinear Programming*. Athena Scientific, Belmont, MA, 1995.

[3] M.M. Deza and M. Laurent. *Geometry of Cuts and Metrics*. Springe-Verlag, 1997.

[4] R. Diestel. *Graph Theory*. Springer-Verlag, 1997.

[5] M.E. Fisher. On the dimer solution of planar ising models. *J. Math. Phys.*, 7:1776–1781, 1966.

[6] M.I. Jordan, editor. *Learning in graphical models*. MIT press, Cambridge, MA, 1998.

[7] P.W. Kasteleyn. Dimer statistics and phase transitions. *Journal of Math. Physics*, 4:287–293, 1963.

[8] L. Lovasz and M.D. Plummer. *Matching Theory*, volume 29 of *Annals of discrete mathematics*. North-Holland, New-York, 1986.

[9] M. J. Wainwright, T. Jaakkola, and A. S. Willsky. Tree-based reparameterization framework for analysis of sum-product and related algorithms. *IEEE Trans. on Information Theory*, 49(5):1120–1146, 2003.

[10] M. J. Wainwright, T. Jaakkola, and A. S. Willsky. Map estimation via agreement on trees: message-passing and linear programming. *IEEE Trans. on Information Theory*, 51(11):1120–1146, 2005.

[11] M. J. Wainwright, T. Jaakkola, and A. S. Willsky. A new class of upper bounds on the log partition function. *IEEE Trans. on Information Theory*, 51(7):2313–2335, 2005.

[12] M.J. Wainwright and M.I. Jordan. Graphical models, exponential families, and variational inference. Technical report, UC Berkeley Dept. of Statistics, 2003.

[13] J.S. Yedidia, W.T. W.T. Freeman, and Y. Weiss. Constructing free-energy approximations and generalized belief propagation algorithms. *IEEE Trans. on Information Theory*, 51(7):2282–2312, 2005.
